# Learning Sparse Multiscale Image Representations

**Phil Sallee**
Department of Computer Science and
Center for Neuroscience, UC Davis
1544 Newton Ct.
Davis, CA 95616
*sallee@cs.ucdavis.edu*

Bruno A. Olshausen
Department of Psychology and
Center for Neuroscience, UC Davis
1544 Newton Ct.
Davis, CA 95616
*baolshausen@ucdavis.edu*

## Abstract

We describe a method for learning sparse multiscale image representations using a sparse prior distribution over the basis function coefficients. The prior consists of a mixture of a Gaussian and a Dirac delta function, and thus encourages coefficients to have exact zero values. Coefficients for an image are computed by sampling from the resulting posterior distribution with a Gibbs sampler. The learned basis is similar to the Steerable Pyramid basis, and yields slightly higher SNR for the same number of active coefficients. Denoising using the learned image model is demonstrated for some standard test images, with results that compare favorably with other denoising methods.

## 1  Introduction

Increasing interest has been given to the use of overcomplete representations for natural scenes, where the number of basis functions exceeds the number of image pixels. One reason for this is that overcompleteness allows for more stable, and thus arguably more meaningful, representations in which common image features can be well described by only a few coefficients, regardless of where they are located in the image, how they are rotated, or how large they are [8, 6]. This may translate into gains in coding efficiency for image compression, and improved accuracy for tasks such as denoising. Overcomplete representations have been shown to reduce Gibbs-like artifacts common to thresholding methods employing critically sampled wavelets [4, 3, 9].

Common wavelet denoising approaches generally apply either a hard or soft-thresholding function to coefficients which have been obtained by filtering an image with a the basis functions. One can view these thresholding methods as a means of selecting coefficients for an image based on an assumed sparse prior on the coefficients [1, 2]. This statistical framework provides a principled means of selecting an appropriate thresholding function. When such thresholding methods are applied to overcomplete representations, however, problems arise due to the dependencies between coefficients. Choosing optimal thresholds for a non-orthogonal basis is still

an unsolved problem. In one approach, orthogonal subgroups of an overcomplete shift-invariant expansion are thresholded separately and then the results are combined by averaging [4, 3]. In addition, if the coefficients are obtained by filtering the noisy image, there will be correlations in the noise that should be taken into account.

Here we address two major issues regarding the use of overcomplete representations for images. First, current methods make use of various overcomplete wavelet bases. What is the optimal basis to use for a specific class of data? To help answer this question, we describe how to adapt an overcomplete wavelet basis to the statistics of natural images. Secondly, we address the problem of properly inferring the coefficients for an image when the basis is overcomplete. We avoid problems associated with thresholding by using the wavelet basis as part of a generative model, rather than a simple filtering mechanism. We then sample the coefficients from the resulting posterior distribution by simulating a Markov process known as a Gibbs-sampler.

Our previous work in this area made use of a prior distribution peaked at zero and tapering away smoothly to obtain sparse coefficients [7]. However, we encountered a number of significant limitations with this method. First, the smooth priors do not force inactive coefficients to have values exactly equal to zero, resulting in decreased coding efficiency. Efficiency may be partially regained by thresholding the near-zero coefficients, but due to the non-orthogonality of the representation this will produce sub-optimal results as previously mentioned. The *maximum a posteriori* (MAP) estimate also introduced biases in the learning process. These effects can be partially compensated for by renormalizing the basis functions, but other parameters of the model such as those of the prior could not be learned. Finally, the gradient ascent method has convergence problems due to the power spectrum of natural images and the overcompleteness of the representation. Here we resolve these problems by using a prior distribution which is composed of a mixture of a Gaussian and a Dirac delta function, so that inactive coefficients are encouraged to have exact zero values. Similar models employing a mixture of two Gaussians have been used for classifying wavelet coefficients into active (high variance) and inactive (low variance) states [2, 5]. Such a classification should be even more advantageous if the basis is overcomplete. A method for performing Gibbs-sampling for the Delta-plus-Gaussian prior in the context of an image pyramid is derived, and demonstrated to be effective at obtaining very sparse representations which match the form of the imposed prior. Biases in the learning are overcome by sampling instead of using a MAP estimate.

## 2   Wavelet image model

Each observed image $\mathbf{I}$ is assumed to be generated by a linear superposition of basis functions which are columns of an $N$ by $M$ weight matrix $\mathbf{W}$, with the addition of Gaussian noise $\nu$:

$$\mathbf{I} = \mathbf{W}\,\mathbf{a} + \nu, \tag{1}$$

where $\mathbf{I}$ is an $N$-element vector of image pixels and $\mathbf{a}$ is an $M$-element vector of basis coefficients. In order to achieve a practical implementation which can be seamlessly scaled to any size image, we assume that the basis function matrix $\mathbf{W}$ is composed of a small set of spatially localized *mother wavelet* functions $\psi_i(x, y)$, which are shifted to each position in the image and rescaled by factors of two. Unlike typical wavelet transforms which use a single 1-D mother wavelet function to generate 2-D functions by inner product, we do not constrain the functions $\psi_i(x, y)$ to be 1-D separable.

The functions $\psi_i(x, y)$ provide an efficient way to perform computations involving $\mathbf{W}$ by means of convolutions. Basis functions of coarser scales are produced by upsampling the $\psi_i(x, y)$ functions and blurring with a low-pass filter $\phi(x, y)$, also known as the *scaling function*. The image model above may be re-expressed to make these parameters explicit:

$$I(x, y) \;=\; g^0(x, y) + \nu(x, y) \tag{2}$$

$$g^l(x, y) \;=\; \begin{cases} [g^{l+1}(x, y) \uparrow 2] * \phi(x, y) + \sum_i a_i^l(x, y) * \psi_i(x, y) & l < L - 1 \\ a^l(x, y) & l = L - 1 \end{cases} \tag{3}$$

where the coefficients $a_i^l(x, y)$ are indexed by their position $(x, y)$, band $(i)$ and level of resolution $(l)$ within the pyramid $(l = 0$ is the highest resolution level). The symbol $*$ denotes convolution, and $\uparrow 2$ denotes upsampling by two and is defined as

$$f(x, y) \uparrow 2 \;\equiv\; \begin{cases} f(\frac{x}{2}, \frac{y}{2}) & x \text{ even } \& \ y \text{ even} \\ 0 & \text{otherwise} \end{cases} \tag{4}$$

The probability of generating an image $\mathbf{I}$, given coefficients $\mathbf{a}$, parameters $\theta$, assuming Gaussian i.i.d. noise $\nu$ (with variance $1/\lambda_N$), is

$$P(\mathbf{I}|\mathbf{a}, \theta) \;=\; \frac{1}{Z_{\lambda_N}} e^{-\frac{\lambda_N}{2}|\mathbf{I} - \mathbf{W}\,\mathbf{a}|^2}. \tag{5}$$

The prior probability over each coefficient $a_i$ is modeled as a mixture of a Gaussian distribution and a Dirac delta function $\delta(a_i)$. A binary state variable $s_i$ for each coefficient indicates whether the coefficient $a_i$ is *active* (any real value), or *inactive* (zero). The probability of a coefficient vector $\mathbf{a}$ given a binary state vector $\mathbf{s}$ and model parameters $\theta = \{\mathbf{W}, \lambda_N, \lambda_{\mathbf{a}}, \mathbf{\Lambda_s}\}$ is defined as

$$P(\mathbf{a}|\mathbf{s}, \theta) \;=\; \prod_i P(a_i|s_i, \theta) \tag{6}$$

$$P(a_i|s_i, \theta) \;=\; \begin{cases} \delta(a_i) & \text{if} \quad s_i = 0, \\ \frac{1}{Z_{\lambda_{a_i}}} e^{-\frac{\lambda_{a_i}}{2} a_i^2} & \text{if} \quad s_i = 1 \end{cases} \tag{7}$$

where $\lambda_{\mathbf{a}}$ is a vector with elements $\lambda_{a_i}$. The probability of a binary state $\mathbf{s}$ is

$$P(\mathbf{s}|\theta) = \frac{1}{Z_{\mathbf{\Lambda_s}}} e^{-\frac{1}{2}\mathbf{s}^T \mathbf{\Lambda_s}\,\mathbf{s}}. \tag{8}$$

Matrix $\mathbf{\Lambda_s}$ is assumed to be diagonal (for now), with nonzero elements $\lambda_{s_i}$. The form of the prior is shown graphically in figure 1. Note that the parameters $\mathbf{W}$, $\lambda_{\mathbf{a}}$, and $\mathbf{\Lambda_s}$ are themselves parameterized by a much smaller set of parameters. Only the mother wavelet function $\psi_i(x, y)$ and a single $\lambda_{s_i}$ and $\lambda_{a_i}$ parameter need to be learned for each wavelet band, since we are assuming translation invariance.

The total image probability is obtained by marginalizing over the possible coefficient and state values:

$$P(\mathbf{I}|\theta) = \sum_{\mathbf{s}} P(\mathbf{s}|\theta) \int P(\mathbf{I}|\mathbf{a}, \theta) P(\mathbf{a}|\mathbf{s}, \theta)\, d\mathbf{a} \tag{9}$$

## 3 Sampling and Inference

We show how to sample from the posterior distribution $P(\mathbf{a}, \mathbf{s}|\mathbf{I}, \theta)$ for an image $\mathbf{I}$ using a Gibbs sampler. For each coefficient and state variable pair $(a_i, s_i)$, we

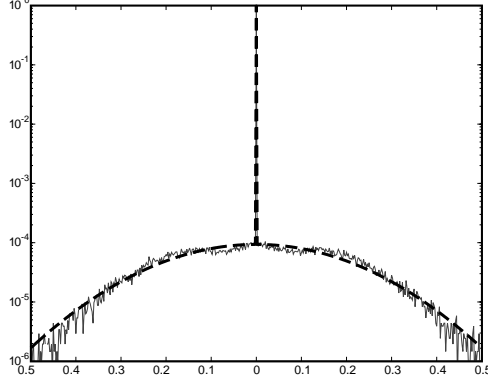

Figure 1: Prior distribution (dashed), and histogram of samples taken from the posterior distribution (solid) plotted for a single coefficient. The y-axis is plotted on a log scale.

sample from the posterior distribution conditioned on the image and the remaining coefficients $a_{\bar{i}}$: $P(a_i, s_i|\mathbf{I}, a_{\bar{i}}, s_{\bar{i}}, \theta)$. After all coefficients (and state variables) have been updated, this process is repeated until the system has reached equilibrium. To infer an optimal representation $\mathbf{a}$ for an image $\mathbf{I}$ (for coding or denoising purposes), we can either average a number of samples to estimate the posterior mean, or with minor adjustment locate a posterior maximum by raising the posterior distribution to a power $(1/T)$ and annealing $T$ to zero. To sample from $P(a_i, s_i|I, a_{\bar{i}}, s_{\bar{i}}, \theta)$, we first draw a value for $s_i$ from $P(s_i|\mathbf{I}, a_{\bar{i}}, s_{\bar{i}}, \theta)$, then draw $a_i$ from $P(a_i|s_i, \mathbf{I}, a_{\bar{i}}, s_{\bar{i}}, \theta)$.

For $P(s_i|\mathbf{I}, a_{\bar{i}}, s_{\bar{i}}, \theta)$ we have:

$$P(s_i|\mathbf{I}, a_{\bar{i}}, s_{\bar{i}}, \theta) \quad \propto \quad P(s_i|s_{\bar{i}}, \theta) \int P(\mathbf{I}|a_i, a_{\bar{i}}, \theta) P(a_i|s_i, \theta) da_i \tag{10}$$

where

$$P(s_i|s_{\bar{i}}, \theta) \quad = \quad \frac{1}{Z_{s_i|s_{\bar{i}}}} e^{-\frac{\lambda_{s_i}}{2} s_i}, \tag{11}$$

$$P(\mathbf{I}|a_i, a_{\bar{i}}, \theta) \quad = \quad \frac{1}{Z_{\lambda_{n_i}}} e^{-\frac{\lambda_{n_i}}{2}(a_i - b_i)^2}, \tag{12}$$

and

$$\lambda_{n_i} = \lambda_N |\mathbf{W}_i|^2, \qquad b_i = \frac{\mathbf{W}_i \cdot (\mathbf{I} - \mathbf{W}\, \mathbf{a}_{i=0})}{|\mathbf{W}_i|^2}. \tag{13}$$

The notation $\mathbf{W}_i$ denotes column $i$ of matrix $\mathbf{W}$, $|\mathbf{W}_i|$ is the length of vector $\mathbf{W}_i$, and $\mathbf{a}_{i=0}$ denotes the current coefficient vector $\mathbf{a}$ except with $a_i$ set to zero. Thus, $b_i$ denotes the value for $a_i$ which minimizes the reconstruction error (while holding $a_{\bar{i}}$ constant). Since $s_i$ can only take on two values, we can compute equation 10 for $s_i = 0$ and $s_i = 1$, integrating over the possible coefficient values. This yields the following sigmoidal activation rule as a function of $b_i$:

$$P(s_i = 1|\mathbf{I}, a_{\bar{i}}, s_{\bar{i}}, \theta) \quad = \quad \frac{1}{1 + e^{-\beta_i(b_i^2 - t_i)}} \tag{14}$$

where

$$\beta_i = \frac{1}{2} \frac{\lambda_{n_i}^2}{\lambda_{n_i} + \lambda_{a_i}}, \qquad t_i = \frac{\lambda_{n_i} + \lambda_{a_i}}{\lambda_{n_i}^2} \left[\lambda_{s_i} - \log \frac{\lambda_{a_i}}{\lambda_{n_i} + \lambda_{a_i}}\right]. \tag{15}$$

For $P(a_i|s_i, \mathbf{I}, a_{\bar{i}}, s_{\bar{i}}, \theta)$ we have:

$$P(a_i|s_i, \mathbf{I}, a_{\bar{i}}, s_{\bar{i}}, \theta) \;\; = \;\; \begin{cases} \delta(a_i) & \text{if} \quad s_i = 0, \\ \mathcal{N}(\frac{\lambda_{n_i} b_i}{\lambda_{n_i} + \lambda_{a_i}}, \frac{1}{\lambda_{n_i} + \lambda_{a_i}}) & \text{if} \quad s_i = 1 \end{cases} \qquad (16)$$

To perform this procedure on a wavelet pyramid, the inner product computations necessary to compute $b_i$ can be performed efficiently by means of convolutions with the mother wavelet functions $\psi_i(x, y)$. The $\lambda_N, \lambda_{s_i}$ and $\lambda_{a_i}$ parameters may be adapted to a specific image during the inference process by use of the update rules described in the next section. This method was found to be particularly useful for denoising, when the variance of the noise was assumed to be unknown.

## 4    Learning

Our objective for learning is to adjust the parameters, $\theta$, to maximize the average log-likelihood of images under the model:

$$\hat{\theta} = \arg \max_{\theta} \langle \log P(\mathbf{I}|\theta) \rangle \qquad (17)$$

The parameters are updated by gradient ascent on this objective, which results in the following update rules:

$$\Delta \lambda_{s_i} \quad \propto \quad \frac{1}{2} \left\langle \left\langle \left[ \frac{1}{1 + e^{\frac{1}{2}\lambda_{s_i}}} - s_i \right] \right\rangle_{P(\mathbf{a},\mathbf{s}|\mathbf{I},\theta)} \right\rangle \qquad (18)$$

$$\Delta \lambda_{a_i} \quad \propto \quad \frac{1}{2} \left\langle \left\langle s_i \left[ \frac{1}{\lambda_{a_i}} - a_i^2 \right] \right\rangle_{P(\mathbf{a},\mathbf{s}|\mathbf{I},\theta)} \right\rangle \qquad (19)$$

$$\Delta \psi_i(x,y) \quad \propto \quad \lambda_N \left\langle \langle e(x,y) \star a_i(x,y) \rangle_{P(\mathbf{a},\mathbf{s}|\mathbf{I},\theta)} \right\rangle \qquad (20)$$

where $\star$ denotes cross correlation and $e(x, y)$ is the reconstruction error computed by $\mathbf{e} = \mathbf{I} - \mathbf{W}\,\mathbf{a}$. Only a center portion of the cross correlation with the extent of the $\psi_i(x, y)$ functions is computed to update the parameters. The outer brackets denotes averaging over many images. The notation $\langle \rangle_{P()}$ denotes averaging the quantity in brackets while sampling from the specified distribution.

## 5    Results

The image model was trained on 22 512x512 pixel grayscale natural images (not whitened). These images were generated from color images taken from a larger database of photographic images [1]. Smaller images (64x64 pixels) were selected randomly for sampling during training. To simplify the learning procedure, sampling was performed on a single spatial frequency scale. Each image was bandpass filtered for an octave range before sampling from the posterior for that scale. The

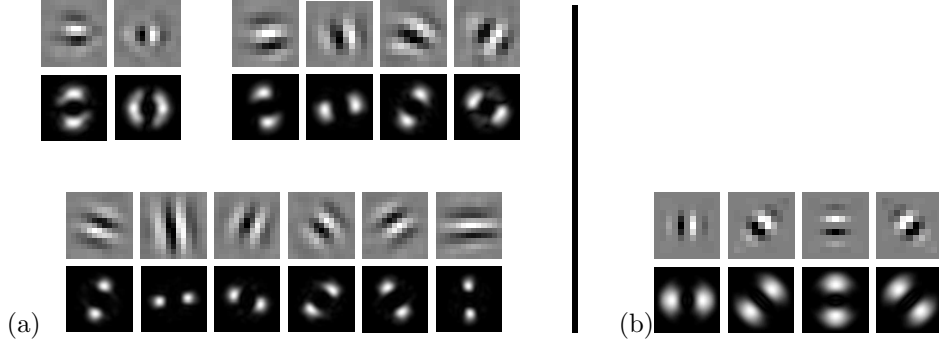

(a)                                                                                    (b)

Figure 2: (a) Mother wavelet functions $\psi_i(x,y)$ adapted for 2, 4 and 6 bands and corresponding power spectra showing power as a function of spatial frequency in the 2D Fourier plane. (b) Equivalent mother wavelets and spectra for the 4-band Steerable Pyramid.

$\lambda_{a_i}$ and $\lambda_{s_i}$ parameters were constrained to be the same for all orientation bands and were adapted over many images with $\lambda_N$ fixed. Shown in figure 2 are the learned $\psi_i(x,y)$ which parameterize $W$, with their corresponding 2D spectra. Three different degrees of overcompleteness were tested. The results are shown for 2 band, 4 band and 6 band wavelet bases. As the degree of overcompleteness increases, the resulting functions show tighter tuning to orientation. The basis filters for a 4 band Steerable Pyramid [10] are also shown for comparison, to illustrate the similarity to the learned functions.

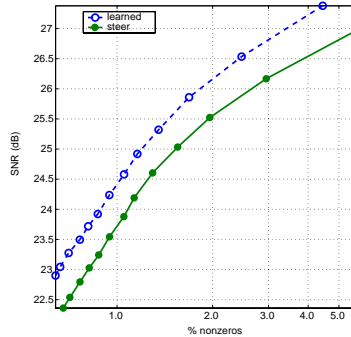

Figure 3: Sparsity comparison between the learned basis (top) and the steerable basis (bottom). The y axis represents the signal-to-noise ratio (SNR) in dB achieved for each method for a given percentage of nonzeros.

## 5.1 Sparsity

We evaluated the sparsity of the representations obtained with the four band learned functions and the sampling method with those obtained using the same sampling method and the four band Steerable Pyramid filters [10]. In order to explore the SNR curves for each basis, we used a variety of values for $\lambda_s$ so as to obtain different levels of sparsity. The same images were used for both bases. The results are given in figure 3. Each dot on the line represents a different value of $\lambda_s$. The results were similar, with the learned basis yielding slightly higher SNR (about 0.5 dB) for the same number of active coefficients.

## 5.2 Denoising

We evaluated our inference method and learned basis functions by denoising images containing known amounts of additive i.i.d. Gaussian noise. Denoising was accomplished by averaging samples taken from the posterior distribution for each image via Gibbs sampling to approximate the posterior mean. Gibbs sampling was performed on a four level pyramid using the 6 band learned wavelet basis, and also using the 6 band Steerable basis. The $\lambda_N, \lambda_{s_i}$ and $\lambda_{a_i}$ parameters were adapted to each noisy image during sampling for blind denoising in which the noise variance was assumed to be unknown. We compared these results to the wiener2 function in MATLAB, and also to BayesCore [9], a Bayesian method for computing an optimal soft thresholding, or coring, function for a generalized Laplacian prior. For wiener2, the best neighborhood size was used for each image. Table 1 gives the SNR results for each method when applied to some standard test images for three different levels of i.i.d. Gaussian noise with standard deviation $\sigma$. Figure 4 shows a cropped subregion of the results for the "Einstein" image with $\sigma = 10$.

## 6 Summary and Conclusions

We have shown that a wavelet basis and a mixture prior composed of a Dirac delta function and a Gaussian can be adapted to natural images resulting in very sparse image representations. The resulting basis is very similar to a Steerable basis, both in appearance and sparsity of the resulting image representations. It appears that the Steerable basis may be nearly optimal for producing sparse representations of natural scenes. Denoising results indicate that using a sparse prior and an inference method to properly account for the non-orthogonality of the representation may yield a significant improvement over wavelet coring methods that use filtered coefficients. More work needs to be done to determine whether the coding gains achieved are due to the choice of prior versus the basis or inference/estimation method used.

**Acknowledgments** Supported by NIMH R29-MH057921. Phil Sallee's work was also supported in part by a United States Department of Education Government Assistance in Areas of National Need (DOE-GAANN) grant #P200A980307.

| Image | noise level | noisy | wiener2 | BayesCore S6 | D+G S6 | D+G L6 |
|---|---|---|---|---|---|---|
| Einstein | $\sigma = 10$ | 12.40 | 15.80 | 16.36 | 16.47 | 16.19 |
| | $\sigma = 20$ | 6.40 | 12.61 | 13.44 | 13.80 | 13.79 |
| | $\sigma = 30$ | 2.89 | 10.95 | 11.81 | 12.28 | 12.29 |
| Lena | $\sigma = 10$ | 13.61 | 19.05 | 19.91 | 20.37 | 20.21 |
| | $\sigma = 20$ | 7.59 | 15.51 | 16.88 | 17.46 | 17.54 |
| | $\sigma = 30$ | 4.07 | 13.25 | 14.99 | 15.48 | 15.55 |
| Goldhill | $\sigma = 10$ | 13.86 | 17.56 | 18.14 | 18.10 | 17.90 |
| | $\sigma = 20$ | 7.83 | 14.32 | 15.18 | 15.41 | 15.41 |
| | $\sigma = 30$ | 4.28 | 12.64 | 13.61 | 13.92 | 13.95 |
| Fruit | $\sigma = 10$ | 16.25 | 21.87 | 22.09 | 22.78 | 22.38 |
| | $\sigma = 20$ | 10.24 | 18.15 | 18.97 | 19.61 | 19.42 |
| | $\sigma = 30$ | 6.70 | 15.97 | 17.21 | 17.72 | 17.66 |

Table 1: SNR values (in dB) for noisy and denoised images contaminated with additive i.i.d. Gaussian noise of std.dev. $\sigma$. "D+G" means Delta-plus-Gaussian prior, "S6" means 6-Band Steerable basis, and "L6" means 6-Band Learned basis.

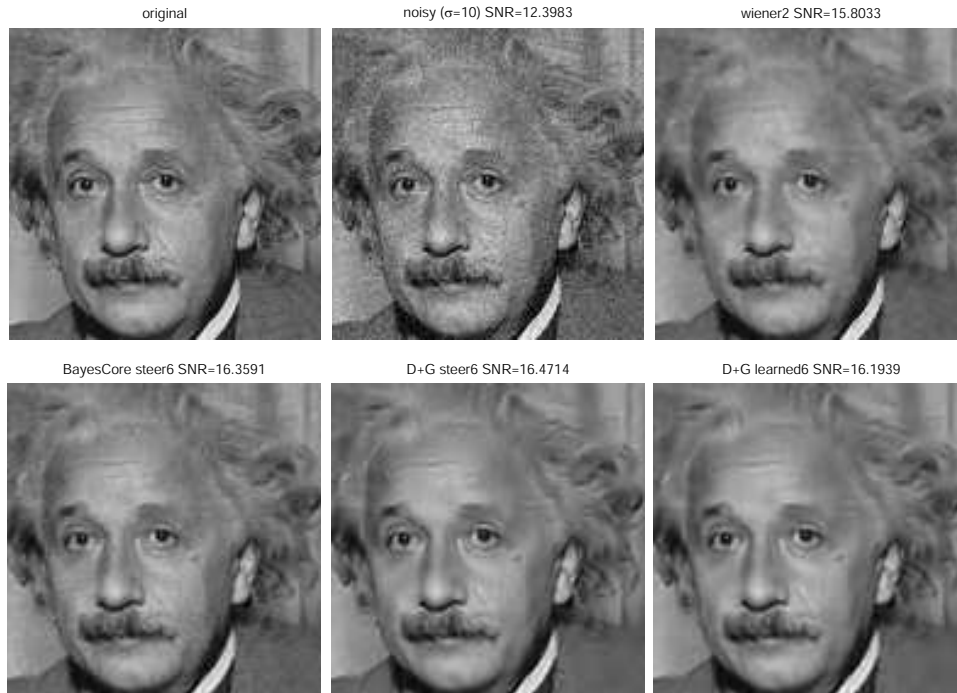

Figure 4: Denoising example. A cropped subregion of the Einstein image and denoised images for each noise reduction method for noise std.dev. $\sigma$=10.

## Footnotes

[1]Images were downloaded from `philip.greenspun.com` with permission from Philip Greenspun.

# References

[1] Abromovich F, Sapatinas T, Silverman B (1996), Wavelet Thresholding via a Bayesian Approach, preprint.

[2] Chipman H, Kolaczyk E, McCulloch R (1997) Adaptive bayesian wavelet shrinkage, *J. Amer. Statist. Assoc.* 92(440): 1413-1421.

[3] Chang SG, Yu B, Vetterli M (2000). Spatially Adaptive Wavelet Thresholding with Context Modelling for Image Denoising. *IEEE Trans. on Image Proc.*, 9(9): 1522-1531.

[4] Coifman RR, Donoho DL (1995). Translation-invariant de-noising, in *Wavelets and Statistics*, A.Antoniadis and G. Oppenheim, Eds. Berlin, Germany: Springer-Varlag.

[5] Crouse MS, Nowak RD and Baraniuk RG (1998) Wavelet-based Statistical Signal Processing using Hidden Markov Models, *IEEE Trans. Signal Proc.*, 46(4): 886-902.

[6] Freeman WT, Adelson EH (1991) The Design and Use of Steerable Filters. *IEEE Trans. Patt. Anal. and Machine Intell.*, 13(9): 891-906.

[7] Olshausen BA, Sallee P, Lewicki MS (2001) Learning sparse image codes using a wavelet pyramid architecture, *Adv. in Neural Inf. Proc. Sys.*, 13: 887-893.

[8] Simoncelli EP, Freeman WT, Adelson EH, Heeger DJ (1992) Shiftable multiscale transforms, *IEEE Transactions on Information Theory*, 38(2): 587-607.

[9] Simoncelli EP, Adelson EH (1996) Noise removal via Bayesian wavelet coring, Presented at: *3rd IEEE International Conf. on Image Proc.*, Laussanne Switzerland.

[10] Simoncelli EP, Freeman WT (1995). The Steerable Pyramid: A Flexible Architecture for Multi-scale Derivative Computation, *IEEE Int. Conf. on Image Processing.*
